# Neuronal Spike Generation Mechanism as an Oversampling, Noise-shaping A-to-D Converter

**Dmitri B. Chklovskii**
Janelia Farm Research Campus
Howard Hughes Medical Institute
mitya@janelia.hhmi.org

**Daniel Soudry**
Department of Electrical Engineering
Technion
daniel.soudry@gmail.com

## Abstract

We test the hypothesis that the neuronal spike generation mechanism is an analog-to-digital (AD) converter encoding rectified low-pass filtered summed synaptic currents into a spike train linearly decodable in post-synaptic neurons. Faithful encoding of an analog waveform by a binary signal requires that the spike generation mechanism has a sampling rate exceeding the Nyquist rate of the analog signal. Such oversampling is consistent with the experimental observation that the precision of the spike-generation mechanism is an order of magnitude greater than the cut-off frequency of low-pass filtering in dendrites. Additional improvement in the coding accuracy may be achieved by noise-shaping, a technique used in signal processing. If noise-shaping were used in neurons, it would reduce coding error relative to Poisson spike generator for frequencies below Nyquist by introducing correlations into spike times. By using experimental data from three different classes of neurons, we demonstrate that biological neurons utilize noise-shaping. Therefore, the spike-generation mechanism can be viewed as an oversampling and noise-shaping AD converter.

The nature of the neural spike code remains a central problem in neuroscience [1-3]. In particular, no consensus exists on whether information is encoded in firing rates [4, 5] or individual spike timing [6, 7]. On the single-neuron level, evidence exists to support both points of view. On the one hand, post-synaptic currents are low-pass-filtered by dendrites with the cut-off frequency of approximately 30Hz [8], Figure 1B, providing ammunition for the firing rate camp: if the signal reaching the soma is slowly varying, why would precise spike timing be necessary? On the other hand, the ability of the spike-generation mechanism to encode harmonics of the injected current up to about 300Hz [9, 10], Figure 1B, points at its exquisite temporal precision [11]. Yet, in view of the slow variation of the somatic current, such precision may seem gratuitous and puzzling.

The timescale mismatch between gradual variation of the somatic current and high precision of spike generation has been addressed previously. Existing explanations often rely on the population nature of the neural code [10, 12]. Although this is a distinct possibility, the question remains whether invoking population coding is necessary. Other possible explanations for the timescale mismatch include the possibility that some synaptic currents (for example, GABAergic) may be generated by synapses proximal to the soma and therefore not subject to low-pass filtering or that the high frequency harmonics are so strong in the pre-synaptic spike that despite attenuation, their trace is still present. Although in some cases, these explanations could apply, for the majority of synaptic inputs to typical neurons there is a glaring mismatch.

The perceived mismatch between the time scales of somatic currents and the spike-generation mechanism can be resolved naturally if one views spike trains as digitally encoding analog somatic currents [13-15], Figure 1A. Although somatic currents vary slowly, information that could be communicated by their analog amplitude far exceeds that of binary signals, such as all-

or-none spikes, of the same sampling rate. Therefore, faithful digital encoding requires sampling rate of the digital signal to be much higher than the cut-off frequency of the analog signal, so-called over-sampling. Although the spike generation mechanism operates in continuous time, the high temporal precision of the spike-generation mechanism may be viewed as a manifestation of oversampling, which is needed for the digital encoding of the analog signal. Therefore, the extra order of magnitude in temporal precision available to the spike-generation mechanism relative to somatic current, Figure 1B, is necessary to faithfully encode the amplitude of the analog signal, thus potentially reconciling the firing rate and the spike timing points of view [13-15].

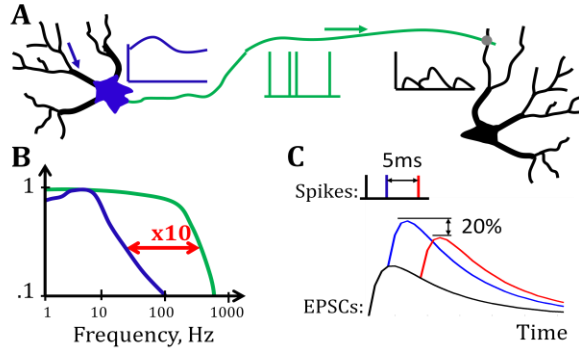

Figure 1. Hybrid digital-analog operation of neuronal circuits. A. Post-synaptic currents are low-pass filtered and summed in dendrites (black) to produce a somatic current (blue). This analog signal is converted by the spike generation mechanism into a sequence of all-or-none spikes (green), a digital signal. Spikes propagate along an axon and are chemically transduced across synapses (gray) into post-synatpic currents (black), whose amplitude reflects synaptic weights, thus converting digital signal back to analog. B. Frequency response function for dendrites (blue, adapted from [8]) and for the spike generation mechanism (green, adapted from [9]). Note one order of magnitude gap between the cut off frequencies. C. Amplitude of the summed post-synaptic currents depends strongly on spike timing. If the blue spike arrives just 5ms later, as shown in red, the EPSCs sum to a value already 20% less. Therefore, the extra precision of the digital signal may be used to communicate the amplitude of the analog signal.

In signal processing, efficient AD conversion combines the principle of oversampling with that of noise-shaping, which utilizes correlations in the digital signal to allow more accurate encoding of the analog amplitude. This is exemplified by a family of AD converters called $\Delta\Sigma$ modulators [16], of which the basic one is analogous to an integrate-and-fire (IF) neuron [13-15]. The analogy between the basic $\Delta\Sigma$ modulator and the IF neuron led to the suggestion that neurons also use noise-shaping to encode incoming analog current waveform in the digital spike train [13]. However, the hypothesis of noise-shaping AD conversion has never been tested experimentally in biological neurons.

In this paper, by analyzing existing experimental datasets, we demonstrate that noise-shaping is present in three different classes of neurons from vertebrates and invertebrates. This lends support to the view that neurons act as oversampling and noise-shaping AD converters and accounts for the mismatch between the slowly varying somatic currents and precise spike timing. Moreover, we show that the degree of noise-shaping in biological neurons exceeds that used by basic $\Delta\Sigma$ modulators or IF neurons and propose viewing more complicated models in the noise-shaping framework. This paper is organized as follows: We review the principles of oversampling and noise-shaping in Section 2. In Section 3, we present experimental evidence for noise-shaping AD conversion in neurons. In Section 4 we argue that rectification of somatic currents may improve energy efficiency and/or implement de-noising.

## 2. Oversampling and noise-shaping in AD converters

To understand how oversampling can lead to more accurate encoding of the analog signal amplitude in a digital form, we first consider a Poisson spike encoder, whose rate of spiking is modulated by the signal amplitude, Figure 2A. Such an AD converter samples an analog signal at discrete time points and generates a spike with a probability given by the (normalized) signal amplitude. Because of the binary nature of spike trains, the resulting spike train encodes the signal with a large error even when the sampling is done at Nyquist rate, i.e. the lowest rate for alias-free sampling.

To reduce the encoding error a Poisson encoder can sample at frequencies, $f_s$, higher than Nyquist, $f_N$ – hence, the term oversampling, Figure 2B. When combined with decoding by low-pass filtering (down to Nyquist) on the receiving end, this leads to a reduction of the error, which can be estimated as follows. The number of samples over a Nyquist half-period ($1/2f_N$) is given by the oversampling ratio:

$$R \sim \frac{f_s}{f_N}.$$

As the normalized signal amplitude, $x \in [0; 1]$, stays roughly constant over the Nyquist half-period, it can be encoded by spikes generated with a fixed probability, $x$. For a Poisson process the variance in the number of spikes is equal to the mean, $\langle (n - \langle n \rangle)^2 \rangle = \langle n \rangle = xR$. Therefore, the mean relative error of the signal decoded by averaging over the Nyquist half-period:

$$e_{rms} = (xR)^{1/2}/R \sim R^{-1/2}, \qquad (1)$$

indicating that oversampling reduces transmission error. However, the weak dependence of the error on the oversampling frequency indicates diminishing returns on the investment in oversampling and motivates one to search for other ways to lower the error.

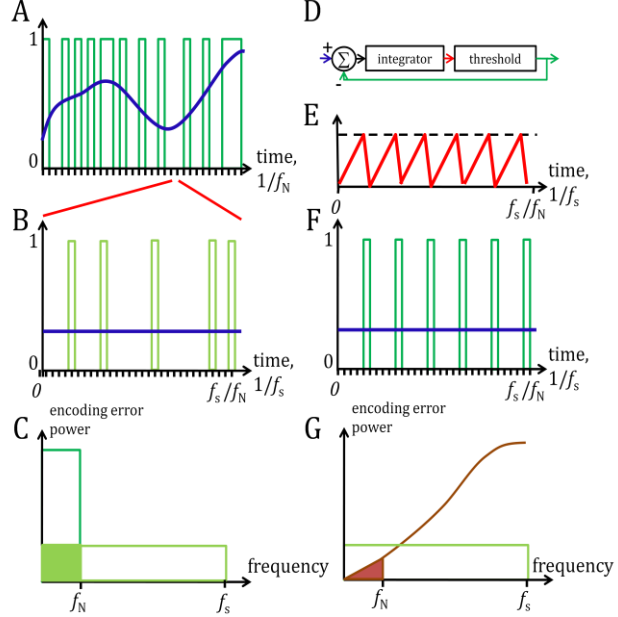

**Figure 2**. **Oversampling and noise-shaping in AD conversion. A**. Analog somatic current (blue) and its digital code (green). The difference between the green and the blue curves is encoding error. **B**. Digital output of oversampling Poisson encoder over one Nyquist half-period. **C**. Error power spectrum of a Nyquist (dark green) and oversampled (light green) Poisson encoder. Although the total error power is the same, the fraction surviving low-pass filtering during decoding (solid green) is smaller in oversampled case. **D**. Basic $\Delta\Sigma$ modulator. **E**. Signal at the output of the integrator. **F**. Digital output of the $\Delta\Sigma$ modulator over one Nyquist period. **G**. Error power spectrum of the $\Delta\Sigma$ modulator (brown) is shifted to higher frequencies and low-pass filtered during decoding. The remaining error power (solid brown) is smaller than for Poisson encoder.

To reduce encoding error beyond the ½ power of the oversampling ratio, the principle of noise-shaping was put forward [17]. To illustrate noise-shaping consider a basic AD converter called $\Delta\Sigma$ [18], Figure 2D. In the basic $\Delta\Sigma$ modulator, the previous quantized signal is fed back and subtracted from the incoming signal and then the difference is integrated in time. Rather than quantizing the input signal, as would be done in the Poisson encoder, $\Delta\Sigma$ modulator quantizes the integral of the difference between the incoming analog signal and the previous quantized signal, Figure 2F. One can see that, in the oversampling regime, the quantization error of the basic $\Delta\Sigma$ modulator is significantly less than that of the Poisson encoder. As the variance in the number of spikes over the Nyquist period is less than one, the mean relative error of the signal is at most, $e_{rms} \sim R^{-1}$, which is better than the Poisson encoder.

To gain additional insight and understand the origin of the term noise-shaping, we repeat the above analysis in the Fourier domain. First, the Poisson encoder has a flat power spectrum up to the sampling frequency, Figure 2C. Oversampling preserves the total error power but extends the frequency range resulting in the lower error power below Nyquist. Second, a more detailed analysis of the basic $\Delta\Sigma$ modulator, where the dynamics is linearized by replacing the quantization device with a random noise injection [19], shows that the quantization noise is effectively differentiated. Taking the derivative in time is equivalent to multiplying the power spectrum of the

quantization noise by frequency squared. Such reduction of noise power at low frequencies is an example of noise shaping, Figure 2G. Under the additional assumption of the white quantization noise, such analysis yields:

$$e_{rms} \sim R^{-3/2}, \qquad (2)$$

which for $R \gg 1$ is significantly better performance than for the Poisson encoder, Eq.(1).

As mentioned previously, the basic $\Delta\Sigma$ modulator, Figure 2D, in the continuous-time regime is nothing other than an IF neuron [13, 20, 21]. In the IF neuron, quantization is implemented by the spike generation mechanism and the negative feedback corresponds to the after-spike reset. Note that resetting the integrator to zero is strictly equivalent to subtraction only for continuous-time operation. In discrete-time computer simulations, the integrator value may exceed the threshold, and, therefore, subtraction of the threshold value rather than reset must be used. Next, motivated by the $\Delta\Sigma$-IF analogy, we look for the signs of noise-shaping AD conversion in real neurons.

## 3. Experimental evidence of noise-shaping AD conversion in real neurons

In order to determine whether noise-shaping AD conversion takes place in biological neurons, we analyzed three experimental datasets, where spike trains were generated by time-varying somatic currents: 1) rat somatosensory cortex L5 pyramidal neurons [9], 2) mouse olfactory mitral cells [22, 23], and 3) fruit fly olfactory receptor neurons [24]. In the first two datasets, the current was injected through an electrode in whole-cell patch clamp mode, while in the third, the recording was extracellular and the intrinsic somatic current could be measured because the glial compartment included only one active neuron.

Testing the noise-shaping AD conversion hypothesis is complicated by the fact that encoded and decoded signals are hard to measure accurately. First, as somatic current is rectified by the spike-generation mechanism, only its super-threshold component can be encoded faithfully making it hard to know exactly what is being encoded. Second, decoding in the dendrites is not accessible in these single-neuron recordings.

In view of these difficulties, we start by simply computing the power spectrum of the reconstruction error obtained by subtracting a scaled and shifted, but otherwise unaltered, spike train from the somatic current. The scaling factor was determined by the total weight of the decoding linear filter and the shift was optimized to maximize information capacity, see below. At the frequencies below 20Hz the error contains significantly lower power than the input signal, Figure 3, indicating that the spike generation mechanism may be viewed as an AD converter. Furthermore, the error power spectrum of the biological neuron is below that of the Poisson encoder, thus indicating the presence of noise-shaping. For dataset 3 we also plot the error power spectrum of the IF neuron, the threshold of which is chosen to generate the same number of spikes as the biological neuron.

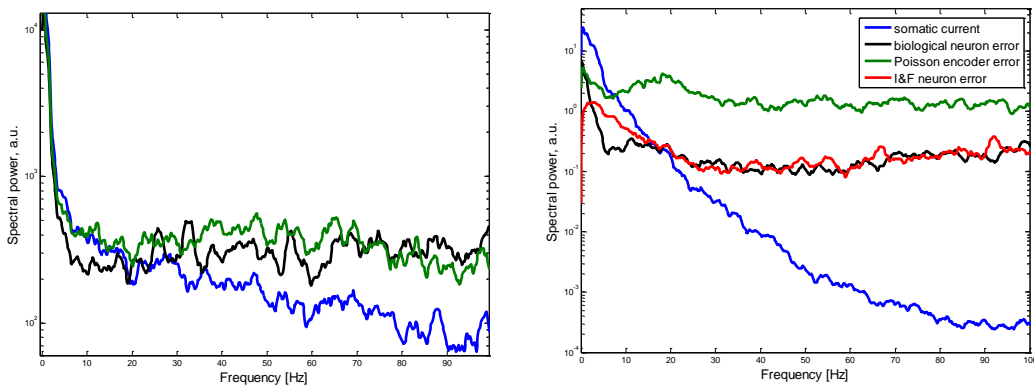

**Figure 3. Evidence of noise-shaping.** Power spectra of the somatic current (blue), difference between the somatic current and the digital spike train of the biological neuron (black), of the Poisson encoder (green) and of the IF neuron (red). Left: datset 1, right: dataset 3.

Although the simple analysis presented above indicates noise-shaping, subtracting the spike train from the input signal, Figure 3, does not accurately quantify the error when decoding involves additional filtering. An example of such additional encoding/decoding is predictive coding, which will be discussed below [25]. To take such decoding filter into account, we computed a decoded waveform by convolving the spike train with the optimal linear filter, which predicts the somatic current from the spike train with the least mean squared error.

Our linear decoding analysis lends additional support to the noise-shaping AD conversion hypothesis [13-15]. First, the optimal linear filter shape is similar to unitary post-synaptic currents, Figure 4B, thus supporting the view that dendrites reconstruct the somatic current of the pre-synaptic neuron by low-pass filtering the spike train in accordance with the noise-shaping principle [13]. Second, we found that linear decoding using an optimal filter accounts for 60-80% of the somatic current variance. Naturally, such prediction works better for neurons in supra-threshold regime, i.e. with high firing rates, an issue to which we return in Section 4. To avoid complications associated with rectification for now we focused on neurons which were in supra-threshold regime by monitoring that the relationship between predicted and actual current is close to linear.

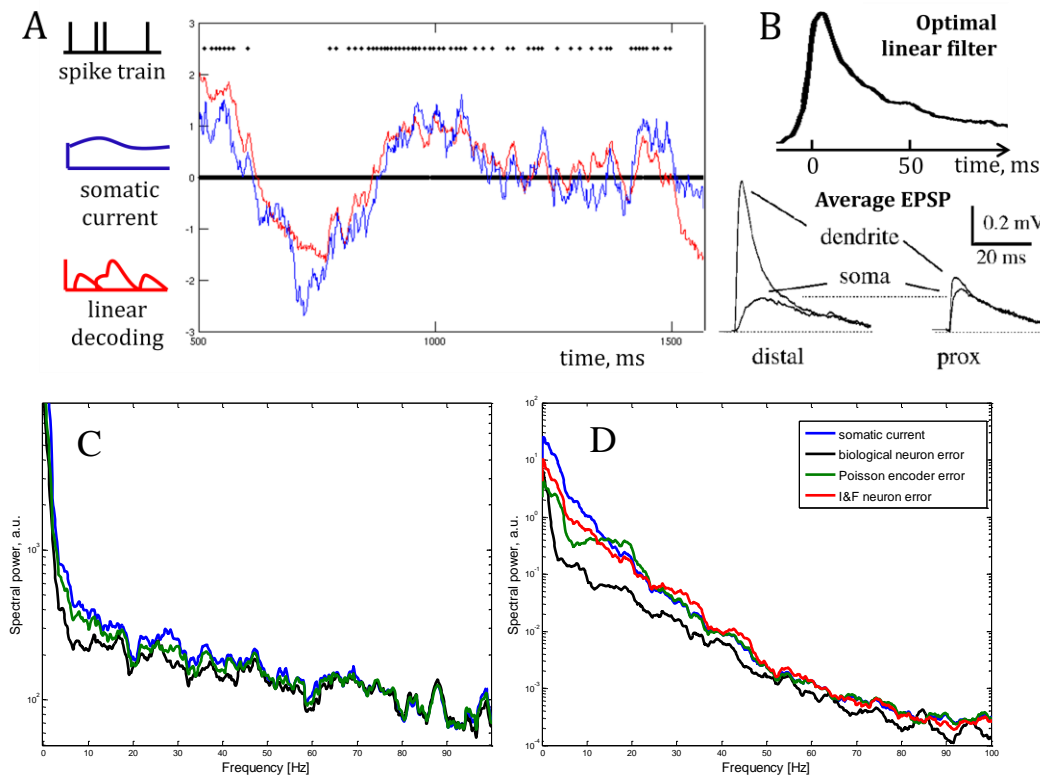

**Figure 4. Linear decoding of experimentally recorded spike trains. A**. Waveform of somatic current (blue), resulting spike train (black), and the linearly decoded waveform (red) from dataset 1. **B**. Top: Optimal linear filter for the trace in **A**, is representative of other datasets as well. Bottom: Typical EPSPs have a shape similar to the decoding filter (adapted from [26]). **C-D**. Power spectra of the somatic current (blue), the decdoding error of the biological neuron (black), the Poisson encoder (green), and IF neuron (red) for dataset 1 (**C**) dataset 3 (**D**).

Next, we analyzed the spectral distribution of the reconstruction error calculated by subtracting the decoded spike train, i.e. convolved with the computed optimal linear filter, from the somatic current. We found that at low frequencies the error power is significantly lower than in the input signal, Figure 4C,D. This observation confirms that signals below the dendritic cut-off frequency of 20-30Hz can be efficiently communicated using spike trains.

To quantify the effect of noise-shaping we computed information capacity of different encoders:

$$I = \sum_f \log\left[\frac{S(f)}{N(f)}\right],$$

where S(f) and N(f) are the power spectra of the somatic current and encoding error correspondingly and the sum is computed only over the frequencies for which S(f) > N(f). Because the plots in Figure 4C,D use semi-logrithmic scale, the information capacity can be estimated from the area between a somatic current (blue) power spectrum and an error power spectrum. We find that the biological spike generation mechanism has higher information capacity than the Poisson encoder and IF neurons. Therefore, neurons act as AD converters with stronger noise-shaping than IF neurons.

We now return to the predictive nature of the spike generation mechanism. Given the causal nature of the spike generation mechanism it is surprising that the optimal filters for all three datasets carry most of their weight following a spike, Figure 4B. This indicates that the spike generation mechanism is capable of making predictions, which are possible in these experiments because somatic currents are temporally correlated. We note that these observations make delay-free reconstruction of the signal possible, thus allowing fast operation of neural circuits [27].

The predictive nature of the encoder can be captured by a $\Delta\Sigma$ modulator embedded in a predictive coding feedback loop [28], Figure 5A. We verified by simulation that such a nested architecture generates a similar optimal linear filter with most of its weight in the time following a spike, Figure 5A right. Of course such prediction is only possible for correlated inputs implying that the shape of the optimal linear filter depends on the statistics of the inputs. The role of predictive coding is to reduce the dynamic range of the signal that enters $\Delta\Sigma$, thus avoiding overloading. A possible biological implementation for such integrating feedback could be $Ca^{2+}$ concentration and $Ca^{2+}$ dependent potassium channels [25, 29].

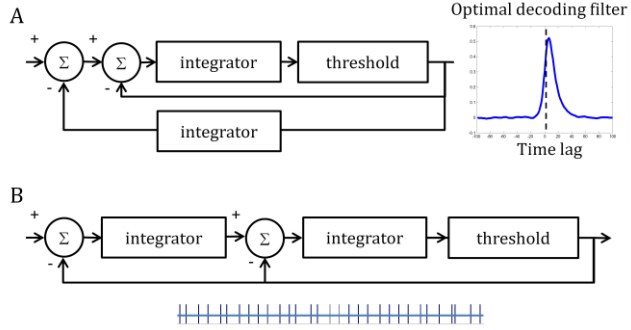

**Figure 5. Enhanced $\Delta\Sigma$ modulators.** A. $\Delta\Sigma$ modulator combined with predictive coder. In such device, the optimal decoding filter computed for correlated inputs has most of its weight following a spike, similar to experimental measurements, Figure 4B. B. Second-order $\Delta\Sigma$ modulator possesses stronger noise-shaping properties. Because such circuit contains an internal state variable it generates a non-periodic spike train in response to a constant input. Bottom trace shows a typical result of a simulation. Black – spikes, blue – input current.

## 4. Possible reasons for current rectification: energy efficiency and de-noising

We have shown that at high firing rates biological neurons encode somatic current into a linearly decodable spike train. However, at low firing rates linear decoding cannot faithfully reproduce the somatic current because of rectification in the spike generation mechanism. If the objective of spike generation is faithful AD conversion, why would such rectification exist? We see two potential reasons: energy efficiency and de-noising.

It is widely believed that minimizing metabolic costs is an important consideration in brain design and operation [30, 31]. Moreover, spikes are known to consume a significant fraction of the metabolic budget [30, 32] placing a premium on their total number. Thus, we can postulate that neuronal spike trains find a trade-off between the mean squared error in the decoded spike train relative to the input signal and the total number of spikes, as expressed by the following cost function over a time interval $T$:

$$s^*_{1..f_sT} = \text{argmin}_s \sum_{t'=1}^{f_sT}(x_{t'} - \sum_{t''} s_{t''} w_{t'-t''})^2 + \lambda \sum_{t'=1}^{f_sT} s_{t'}, \qquad (3)$$

where $x$ is the analog input signal, $s$ is the binary spike sequence composed of zeros and ones, and $w$ is the linear filter.

To demonstrate how solving Eq.(3) would lead to thresholding, let us consider a simplified version taken over a Nyquist period, during which the input signal stays constant:

$$N_s = \text{argmin}_{N=0,1,2,...}(\bar{x} - N)^2 + \bar{\lambda}N \qquad (4)$$

where $\bar{x}$ and $\bar{\lambda}$ normalized by $w$. Minimizing such a cost function reduces to choosing the lowest lying parabola for a given $\bar{x}$, Figure 6A. Therefore, thresholding is a natural outcome of minimizing a cost function combining the decoding error and the energy cost, Eq.(3).

In addition to energy efficiency, there may be a computational reason for thresholding somatic current in neurons. To illustrate this point, we note that the cost function in Eq. (3) for continuous variables, $s_t$, may be viewed as a non-negative version of the L1-norm regularized linear regression called LASSO [33], which is commonly used for de-noising of sparse and Laplacian signals [34]. Such cost function can be minimized by iteratively applying a gradient descent and a shrinkage steps [35], which is equivalent to thresholding (one-sided in case of non-negative variables), Figure 6B,C. Therefore, neurons may be encoding a de-noised input signal.

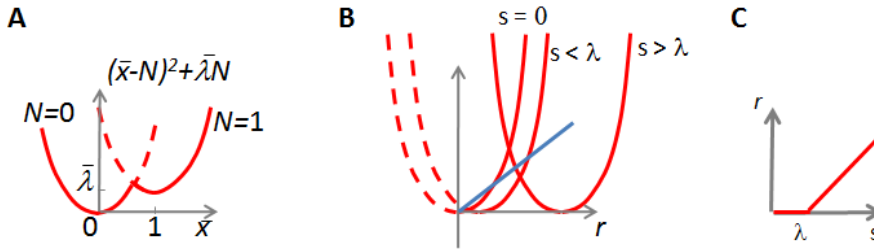

**Figure 6. Possible reasons for rectification in neurons. A.** Cost function combining encoding error squared with metabolic expense vs. input signal $\bar{x}$ for different values of the spike number $N$, Eq.(4). Note that the optimal number of spikes jumps from zero to one as a function of input. **B.** Estimating most probable "clean" signal value for continuous non-negative Laplacian signal and Gaussian noise, Eq.(3) (while setting $w = 1$). The parabolas (red) illustrate the quadratic log-likelihood term in (3) for different values of the measurement, s, while the linear function (blue) reflects the linear log-prior term in (3). **C.** The minimum of the combined cost function in **B** is at zero if $s < \lambda$, and grows linearly with s, if $s > \lambda$.

## 5. Discussion

In this paper, we demonstrated that the neuronal spike-generation mechanism can be viewed as an oversampling and noise-shaping AD converter, which encodes a rectified low-pass filtered somatic current as a digital spike train. Rectification by the spike generation mechanism may subserve both energy efficiency and de-noising. As the degree of noise-shaping in biological neurons exceeds that in IF neurons, or basic $\Delta\Sigma$, we suggest that neurons should be modeled by more advanced $\Delta\Sigma$ modulators, e.g. Figure 5B. Interestingly, $\Delta\Sigma$ modulators can be also viewed as coders with error prediction feedback [19].

Many publications studied various aspects of spike generation in neurons yet we believe that the framework [13-15] we adopt is different and discuss its relationship to some of the studies. Our framework is different from previous proposals to cast neurons as predictors [36, 37] because a different quantity is being predicted. The possibility of perfect decoding from a spike train with infinite temporal precision has been proven in [38]. Here, we are concerned with a more practical issue of how reconstruction error scales with the over-sampling ratio. Also, we consider linear decoding which sets our work apart from [39]. Finally, previous experiments addressing noise-shaping [40] studied the power spectrum of the spike train rather than that of the encoding error.

Our work is aimed at understanding biological and computational principles of spike-generation and decoding and is not meant as a substitute for the existing phenomenological spike-generation models [41], which allow efficient fitting of parameters and prediction of spike trains [42]. Yet, the theoretical framework [13-15] we adopt may assist in building better models of spike generation for a given somatic current waveform. First, having interpreted spike generation as AD conversion, we can draw on the rich experience in signal processing to attack the problem. Second, this framework suggests a natural metric to compare the performance of different spike generation models in the high firing rate regime: a mean squared error between the injected

current waveform and the filtered version of the spike train produced by a model provided the total number of spikes is the same as in the experimental data. The AD conversion framework adds justification to the previously proposed spike distance obtained by subtracting low-pass filtered spike trains [43].

As the framework [13-15] we adopt relies on viewing neuronal computation as an analog-digital hybrid, which requires AD and DA conversion at every step, one may wonder about the reason for such a hybrid scheme. Starting with the early days of computers, the analog mode is known to be advantageous for computation. For example, performing addition of many variables in one step is possible in the analog mode simply by Kirchhoff law, but would require hundreds of logical gates in the digital mode [44]. However, the analog mode is vulnerable to noise build-up over many stages of computation and is inferior in precisely communicating information over long distances under limited energy budget [30, 31]. While early analog computers were displaced by their digital counterparts, evolution combined analog and digital modes into a computational hybrid [44], thus necessitating efficient AD and DA conversion, which was the focus of the present study.

We are grateful to L. Abbott, S. Druckmann, D. Golomb, T. Hu, J. Magee, N. Spruston, B. Theilman for helpful discussions and comments on the manuscript, to X.-J. Wang, D. McCormick, K. Nagel, R. Wilson, K. Padmanabhan, N. Urban, S. Tripathy, H. Koendgen, and M. Giugliano for sharing their data. The work of D.S. was partially supported by the Intel Collaborative Research Institute for Computational Intelligence (ICRI-CI).

## References

1. Ferster, D. and N. Spruston, *Cracking the neural code.* Science, 1995. **270**: p. 756-7.
2. Panzeri, S., et al., *Sensory neural codes using multiplexed temporal scales.* Trends Neurosci, 2010. **33**(3): p. 111-20.
3. Stevens, C.F. and A. Zador, *Neural coding: The enigma of the brain.* Curr Biol, 1995. **5**(12): p. 1370-1.
4. Shadlen, M.N. and W.T. Newsome, *The variable discharge of cortical neurons: implications for connectivity, computation, and information coding.* J Neurosci, 1998. **18**(10): p. 3870-96.
5. Shadlen, M.N. and W.T. Newsome, *Noise, neural codes and cortical organization.* Curr Opin Neurobiol, 1994. **4**(4): p. 569-79.
6. Singer, W. and C.M. Gray, *Visual feature integration and the temporal correlation hypothesis.* Annu Rev Neurosci, 1995. **18**: p. 555-86.
7. Meister, M., *Multineuronal codes in retinal signaling.* Proc Natl Acad Sci U S A, 1996. **93**(2): p. 609-14.
8. Cook, E.P., et al., *Dendrite-to-soma input/output function of continuous time-varying signals in hippocampal CA1 pyramidal neurons.* J Neurophysiol, 2007. **98**(5): p. 2943-55.
9. Kondgen, H., et al., *The dynamical response properties of neocortical neurons to temporally modulated noisy inputs in vitro.* Cereb Cortex, 2008. **18**(9): p. 2086-97.
10. Tchumatchenko, T., et al., *Ultrafast population encoding by cortical neurons.* J Neurosci, 2011. **31**(34): p. 12171-9.
11. Mainen, Z.F. and T.J. Sejnowski, *Reliability of spike timing in neocortical neurons.* Science, 1995. **268**(5216): p. 1503-6.
12. Mar, D.J., et al., *Noise shaping in populations of coupled model neurons.* Proc Natl Acad Sci U S A, 1999. **96**(18): p. 10450-5.
13. Shin, J., *Adaptive noise shaping neural spike encoding and decoding.* Neurocomputing, 2001. **38-40**: p. 369-381.
14. Shin, J., *The noise shaping neural coding hypothesis: a brief history and physiological implications.* Neurocomputing, 2002. **44**: p. 167-175.
15. Shin, J.H., *Adaptation in spiking neurons based on the noise shaping neural coding hypothesis.* Neural Networks, 2001. **14**(6-7): p. 907-919.
16. Schreier, R. and G.C. Temes, *Understanding delta-sigma data converters* 2005, Piscataway, NJ: IEEE Press, Wiley. xii, 446 p.
17. Candy, J.C., *A use of limit cycle oscillations to obtain robust analog-to-digital converters.* IEEE Trans. Commun, 1974. **COM-22**: p. 298-305.

18. Inose, H., Y. Yasuda, and J. Murakami, *A telemetring system code modulation - ΔΣ modulation.* IRE Trans. Space Elect. Telemetry, 1962. **SET-8**: p. 204-209.

19. Spang, H.A. and P.M. Schultheiss, *Reduction of quantizing noise by use of feedback.* IRE TRans. Commun. Sys., 1962: p. 373-380.

20. Hovin, M., et al., *Delta-Sigma modulation in single neurons*, in *IEEE International Symposium on Circuits and Systems*2002.

21. Cheung, K.F. and P.Y.H. Tang, *Sigma-Delta Modulation Neural Networks.* Proc. IEEE Int Conf Neural Networkds, 1993: p. 489-493.

22. Padmanabhan, K. and N. Urban, *Intrinsic biophysical diversity decorelates neuronal firing while increasing information content.* Nat Neurosci, 2010. **13**: p. 1276-82.

23. Urban, N. and S. Tripathy, *Neuroscience: Circuits drive cell diversity.* Nature, 2012. **488**(7411): p. 289-90.

24. Nagel, K.I. and R.I. Wilson, *personal communication*.

25. Shin, J., C. Koch, and R. Douglas, *Adaptive neural coding dependent on the time-varying statistics of the somatic input current.* Neural Comp, 1999. **11**: p. 1893-913.

26. Magee, J.C. and E.P. Cook, *Somatic EPSP amplitude is independent of synapse location in hippocampal pyramidal neurons.* Nat Neurosci, 2000. **3**(9): p. 895-903.

27. Thorpe, S., D. Fize, and C. Marlot, *Speed of processing in the human visual system.* Nature, 1996. **381**(6582): p. 520-2.

28. Tewksbury, S.K. and R.W. Hallock, *Oversample, linear predictive and noise-shaping coders of order N>1.* IEEE Trans Circuits & Sys, 1978. **CAS25**: p. 436-47.

29. Wang, X.J., et al., *Adaptation and temporal decorrelation by single neurons in the primary visual cortex.* J Neurophysiol, 2003. **89**(6): p. 3279-93.

30. Attwell, D. and S.B. Laughlin, *An energy budget for signaling in the grey matter of the brain.* J Cereb Blood Flow Metab, 2001. **21**(10): p. 1133-45.

31. Laughlin, S.B. and T.J. Sejnowski, *Communication in neuronal networks.* Science, 2003. **301**(5641): p. 1870-4.

32. Lennie, P., *The cost of cortical computation.* Curr Biol, 2003. **13**(6): p. 493-7.

33. Tibshirani, R., *Regression shrinkage and selection via the Lasso.* Journal of the Royal Statistical Society Series B-Methodological, 1996. **58**(1): p. 267-288.

34. Chen, S.S.B., D.L. Donoho, and M.A. Saunders, *Atomic decomposition by basis pursuit.* Siam Journal on Scientific Computing, 1998. **20**(1): p. 33-61.

35. Elad, M., et al., *Wide-angle view at iterated shrinkage algorithms.* P SOc Photo-Opt Ins, 2007. **6701**: p. 70102.

36. Deneve, S., *Bayesian spiking neurons I: inference.* Neural Comp, 2008. **20**: p. 91.

37. Yu, A.J., *Optimal Change-Detection and Spiking Neurons*, in *NIPS*, B. Scholkopf, J. Platt, and T. Hofmann, Editors. 2006.

38. Lazar, A. and L. Toth, *Perfect Recovery and Sensitivity Analysis of Time Encoded Bandlimited Signals.* IEEE TRANSACTIONS ON CIRCUITS AND SYSTEMS, 2004. **51**(10).

39. Pfister, J.P., P. Dayan, and M. Lengyel, *Synapses with short-term plasticity are optimal estimators of presynaptic membrane potentials.* Nat Neurosci, 2010. **13**(10): p. 1271-5.

40. Chacron, M.J., et al., *Experimental and theoretical demonstration of noise shaping by interspike interval correlations.* Fluctuations and Noise in Biological, Biophysical, and Biomedical Systems III, 2005. **5841**: p. 150-163.

41. Pillow, J., *Likelihood-based approaches to modeling the neural code*, in *Bayesian Brain: Probabilistic Approaches to Neural Coding*, K. Doya, et al., Editors. 2007, MIT Press.

42. Jolivet, R., et al., *A benchmark test for a quantitative assessment of simple neuron models.* J Neurosci Methods, 2008. **169**(2): p. 417-24.

43. van Rossum, M.C., *A novel spike distance.* Neural Comput, 2001. **13**(4): p. 751-63.

44. Sarpeshkar, R., *Analog versus digital: extrapolating from electronics to neurobiology.* Neural Computation, 1998. **10**(7): p. 1601-38.
